# Scan Strategies for Adaptive Meteorological Radars

**Victoria Manfredi, Jim Kurose**
Department of Computer Science
University of Massachusetts
Amherst, MA USA
{vmanfred,kurose}@cs.umass.edu

## Abstract

We address the problem of adaptive sensor control in dynamic resource-constrained sensor networks. We focus on a meteorological sensing network comprising radars that can perform sector scanning rather than always scanning $360°$. We compare three sector scanning strategies. The sit-and-spin strategy always scans $360°$. The limited lookahead strategy additionally uses the expected environmental state $K$ decision epochs in the future, as predicted from Kalman filters, in its decision-making. The full lookahead strategy uses all expected future states by casting the problem as a Markov decision process and using reinforcement learning to estimate the optimal scan strategy. We show that the main benefits of using a lookahead strategy are when there are multiple meteorological phenomena in the environment, and when the maximum radius of any phenomenon is sufficiently smaller than the radius of the radars. We also show that there is a trade-off between the average quality with which a phenomenon is scanned and the number of decision epochs before which a phenomenon is rescanned.

## 1   Introduction

Traditionally, meteorological radars, such as the National Weather Service NEXRAD system, are tasked to always scan 360 degrees. In contrast, the Collaborative Adaptive Sensing of the Atmosphere (CASA) Engineering Research Center [5] is developing a new generation of small, low-power but agile radars that can perform sector scanning, targeting sensing when and where the user needs are greatest. Since all meteorological phenomena cannot be now all observed all of the time with the highest degree of fidelity, the radars must decide how best to perform scanning. While we focus on the problem of how to perform sector scanning in such an adaptive meteorological sensing network, it is an instance of the larger class of problems of adaptive sensor control in dynamic resource-constrained sensor networks.

Given the ability of a network of radars to perform sector scanning, how should scanning be adapted at each decision epoch? Any scan strategy must consider, for each scan action, both the expected quality with which phenomena would be observed, and the expected number of decision epochs before which phenomena would be first observed (for new phenomena) or rescanned, since not all regions are scanned every epoch under sectored scanning. Another consideration is whether to optimize myopically only over current and possibly past environmental state, or whether to additionally optimize over expected future states. In this work we examine three methods for adapting the radar scan strategy. The methods differ in the information they use to select a scan configuration at a particular decision epoch. The sit-and-spin strategy of always scanning 360 degrees is independent of any external information. The limited lookahead strategies additionally use the expected environmental state $K$ decision epochs in the future in its decision-making. Finally, the full lookahead strategy has an infinite horizon: it uses all expected future states by casting the problem as a Markov decision process and using reinforcement learning to estimate the optimal scan strategy. All strategies, excluding sit-and-spin, work by optimizing the overall "quality" (a term we will define

precisely shortly) of the sensed information about phenomena in the environment, while restricting or penalizing long inter-scan intervals.

Our contributions are two-fold. We first introduce the meteorological radar control problem and show how to constrain the problem so that it is amenable to reinforcement learning methods. We then identify conditions under which the computational cost of an infinite horizon radar scan strategy such as reinforcement learning is necessary. With respect to the radar meteorological application, we show that the main benefits of considering expected future states are when there are multiple meteorological phenomena in the environment, and when the maximum radius of any phenomenon is sufficiently smaller than the radius of the radars. We also show that there is a trade-off between the average quality with which a phenomenon is scanned and the number of decision epochs before which a phenomenon is rescanned. Finally, we show that for some environments, a limited looka-head strategy is sufficient. In contrast to other work on radar control (see Section 5), we focus on tracking meteorological phenomena and the time frame over which to evaluate control decisions.

The rest of this paper is organized as follows. Section 2 defines the radar control problem. Section 3 describes the scan strategies we consider. Section 4 describes our evaluation framework and presents results. Section 5 reviews related work on control and resource allocation in radar and sensor networks. Finally, Section 6 summarizes this work and outlines future work.

## 2   Meteorological Radar Control Problem

Meteorological radar sensing characteristics are such that the smaller the sector that a radar scans (until a minimum sector size is reached), the higher the quality of the data collected, and thus, the more likely it is that phenomena located within the sector are correctly identified [2]. The multi-radar meteorological control problem is then as follows. We have a set of radars, with fixed locations and possibly overlapping footprints. Each radar has a set of scan actions from which it chooses. In the simplest case, a radar scan action determines the size of the sector to scan, the start angle, the end angle, and the angle of elevation. We will not consider elevation angles here. Our goal is to determine which scan actions to use and when to use them. An effective scanning strategy must balance scanning small sectors (thus implicitly *not* scanning other sectors), to ensure that phenomena are correctly identified, with scanning a variety of sectors, to ensure that no phenomena are missed.

We will evaluate the performance of different scan strategies based on inter-scan time, quality, and cost. Inter-scan time is the number of decision epochs before a phenomenon is either first observed or rescanned; we would like this value to be below some threshold. Quality measures how well a phenomenon is observed, with quality depending on the amount of time a radar spends sampling a voxel in space, the degree to which a meteorological phenomena is scanned in its (spatial) entirety, and the number of radars observing a phenomenon; higher quality scans are better. Cost is a meta-metric that combines inter-scan time and quality, and that additionally considers whether a phenomenon was never scanned. The radar control problem is that of dynamically choosing the scan strategy of the radars over time to maximize quality while minimizing inter-scan time.

## 3   Scan Strategies

We define a *radar configuration* to be the start and end angles of the sector to be scanned by an individual radar for a fixed interval of time. We define a *scan action* to be a set of radar configurations (one configuration for each radar in the meteorological sensing network). We define a *scan strategy* to be an algorithm for choosing scan actions. In Section 3.1 we define the quality function associated with different radar configurations and in Section 3.2 we define the quality functions associated with different scan strategies.

### 3.1   Quality Function

The quality function associated with a given scan action was proposed by radar meteorologists in [5] and has two components. There is a quality component $U_p$ associated with scanning a particular phenomenon $p$. There is also a quality component $U_s$ associated with scanning a sector, which is independent of any phenomena in that sector. Let $s_r$ be the radar configuration for a single radar $r$ and let $S_r$ be the scan action under consideration. From [5], we compute the quality $U_p(p, S_r)$ of

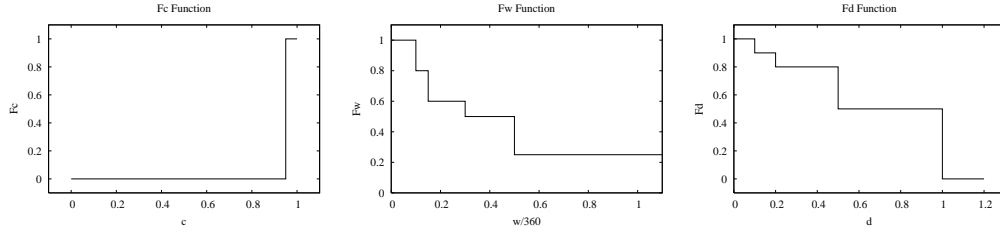

Figure 1: Step functions used by the $U_p$ and $U_s$ quality functions, from [9]

scanning a phenomenon $p$ using scan action $S_r$ with the following equations,

$$U_p(p, s_r) = F_c\left(c(p, s_r)\right) \times \left[\beta F_d\left(d(r, p)\right) + (1 - \beta)F_w\left(\frac{w(s_r)}{360}\right)\right]$$
$$U_p(p, S_r) = \max_{s_r \in S_r}\left[U_p(p, s_r)\right] \tag{1}$$

where

$$
\begin{aligned}
w(s_r) &= \text{size of sector } s_r \text{ scanned by } r \\
a(r, p) &= \text{minimal angle that would allow } r \text{ to cover } p \\
c(p, s_r) &= \frac{w(s_r)}{a(r, p)} = \text{coverage of } p \text{ by } r \text{ scanning } s_r \\
h(r, p) &= \text{distance from } r \text{ to geometric center of } p \\
h_{max}(r) &= \text{range of radar } r \\
d(r, p) &= \frac{h(r, p)}{h_{max}(r)} = \text{normalized distance from } r \text{ to } p \\
\beta &= \text{tunable parameter}
\end{aligned}
$$

$U_p(p, S_r)$ is the maximum quality obtained for scanning phenomenon $p$ over all possible radars and their associated radar configurations $s_r$. $U_p(p, s_r)$ is the quality obtained for scanning phenomenon $p$ using a specific radar $r$ and radar configuration $s_r$. The functions $F_c(\cdot)$, $F_w(\cdot)$, and $F_d(\cdot)$ from [5] are plotted in Figure 1. $F_c$ captures the effect on quality due to the percentage of the phenomenon covered; to usefully scan a phenomenon, at least 95% of the phenomenon must be scanned. $F_w$ captures the effect of radar rotation speed on quality; as rotation speed is reduced, quality increases. $F_d$ captures the effects of the distance from the radar to the geometrical center of the phenomenon on quality; the further away the radar center is from the phenomenon being scanned, the more degraded will be the scan quality due to attenuation. Due to the $F_w$ function, the quality function $U_p(p, s_r)$ outputs the same quality for scan angles of $181°$ to $360°$. The quality $U_s(r_i, s_r)$ for scanning a subsector $i$ of radar $r$ scanned using configuration $s_r$ is,

$$U_s(r_i, s_r) = F_w\left(\frac{w(s_r)}{360}\right) \tag{2}$$

Intuitively, a sector scanning strategy is only preferable when the quality function is such that the quality gained for scanning a sector is greater than the quality lost for not scanning another sector.

### 3.2 Scan Strategies

We compare the performance of the following three scan strategies. The strategies differ in whether they optimize quality over only current or also future expected states. For example, suppose a storm cell is about to move into a high-quality multi-doppler region (i.e., the area where multiple radar footprints overlap). By considering future expected states, a lookahead strategy can anticipate this event and have all radars focused on the storm cell when it enters the multi-doppler region, rather than expending resources (with little "reward") to scan the storm cell just before it enters this region.

*(i) Sit-and-spin strategy.* All radars always scan $360°$.

*(ii) Limited "lookahead" strategy.* We examine both a 1-step and a 2-step look-ahead scan strategy. Although we do not have an exact model of the dynamics of different phenomena, to perform the

look-ahead we estimate the future attributes of each phenomenon using a separate Kalman filter. For each filter, the true state $\mathbf{x}$ is a vector comprising the $(x, y)$ location and velocity of the phenomenon, and the measurement $\mathbf{y}$ is a vector comprising only the $(x, y)$ location. The Kalman filter assumes that the state at time $t$ is a linear function of the state at time $t - 1$ plus some Gaussian noise, and that the measurement at time $t$ is a linear function of the state at time $t$ plus some Gaussian noise. In particular, $\mathbf{x}_t = \mathbf{A}\mathbf{x}_{t-1} + N[\mathbf{0}, \mathbf{Q}]$ and $\mathbf{y}_t = \mathbf{B}\mathbf{x}_t + N[\mathbf{0}, \mathbf{R}]$.

Following work by [8], we initialize each Kalman filter as follows. The $\mathbf{A}$ matrix reflects that storm cells typically move to the north-east. The $\mathbf{B}$ matrix, which when multiplied with $\mathbf{x}_t$ returns $\mathbf{x}_t$, assumes that the observed state $\mathbf{y}_t$ is directly the true state $\mathbf{x}_t$ plus some Gaussian noise. The $\mathbf{Q}$ matrix assumes that there is little noise in the true state dynamics. Finally, the measurement error covariance matrix $\mathbf{R}$ is a function of the quality $U_p$ with which phenomenon $p$ was scanned at time $t$. We discuss how to compute the $\sigma_t$'s in Section 4. We use the first location measurement of a storm cell $\mathbf{y}_0$, augmented with the observed velocity, as the the initial state $\mathbf{x}_0$. We assume that our estimate of $\mathbf{x}_0$ has little noise and use $.0001 * I$ for the initial covariance $\mathbf{P}_0$.

$$
\mathbf{A} = \begin{bmatrix} 1 & 0 & 1 & 0 \\ 0 & 1 & 0 & 1 \\ 0 & 0 & 1 & 0 \\ 0 & 0 & 0 & 1 \end{bmatrix}, \mathbf{B} = \begin{bmatrix} 1 & 0 & 0 & 0 \\ 0 & 1 & 0 & 0 \end{bmatrix}, \mathbf{Q} = \begin{bmatrix} .0001 & 0 & 0 & 0 \\ 0 & .0001 & 0 & 0 \\ 0 & 0 & .0001 & 0 \\ 0 & 0 & 0 & .0001 \end{bmatrix}, \mathbf{R} = \begin{bmatrix} \sigma_t & 0 \\ 0 & \sigma_t \end{bmatrix}
$$

We compute the $k$-step look-ahead quality for different sets of radar configurations $S_r$ with,

$$
U_K(S_{r,1}|T_r) \quad = \quad \sum_{k=1}^{K} \phi^{k-1} \sum_{i=1}^{N_p} U_p(p_{i,k}, S_{r,k}|T_r)
$$

where $N_p$ is the number of phenomena in the environment in the current decision epoch, $p_{i,0}$ is the current set of observed attributes for phenomenon $i$, $p_{i,k}$ is the $k$-step set of predicted attributes for phenomenon $i$, $S_{r,k}$ is the set of radar configurations for the $k$th decision epoch in the future, and $\phi$ is a tunable discount factor between 0 and 1. The optimal set of radar configurations is then $S_{r,1}^* = \text{argmax}_{S_{r,1}} U_K(S_{r,1}|T_r)$. To account for the decay of quality for unscanned sectors and phenomena, and to consider the possibility of new phenomena appearing, we restrict $S_r$ to be those scan actions that ensure that every sector has been scanned at least once in the last $T_r$ decision epochs. $T_r$ is a tunable parameter whose purpose is to satisfy the meteorological dictate found in [5], that all sectors be scanned, for instance by a $360°$ scan, at most every 5 minutes.

*(iii) Full "lookahead" strategy.* We formulate the radar control problem as a Markov decision process (MDP) and use reinforcement learning to obtain a lookahead scan strategy as follows. While a POMDP (partially observable MDP) could be used to model the environmental uncertainty, due to the cost of solving a POMDP with a large state space [9], we choose to formulate the radar control problem as an MDP with quality (or uncertainty) variables as in an augmented MDP [6].

*S is the observed state of the environment.* The state is a function of the observed number of storms, the observed $x, y$ velocity of each storm, and the observed dimensions of each storm cell given by $x, y$ center of mass and radius. To model the uncertainty in the environment, we additionally define as part of the state quality variables $u_p$ and $u_s$ based on the $U_p$ and $U_s$ quality functions defined in Equations (1) and (2) in Section 3.1. $u_p$ is the quality $U_p(\cdot)$ with which each storm cell was observed, and $u_s$ is the current quality $U_s(\cdot)$ of each $90°$ subsector, starting at 0, 90, 180, or $270°$.

*A is the set of actions available to the radars.* This is the set of radar configurations for a given decision epoch. We restrict each radar to scanning subsectors that are a multiple of $90°$, starting at 0, 90, 180, or $270°$. Thus, with N radars there are $13^N$ possible actions at each decision epoch.

*The transition function $T(S \times A \times S) \to [0, 1]$* encodes the *observed* environment dynamics: specifically the appearance, disappearance, and movement of storm cells and their associated attributes. For meteorological radar control, the next state really is a function of not just the current state but also the action executed in the current state. For instance, if a radar scans 180 degrees rather than 360 degrees, then any new storm cells that appear in the unscanned areas will not be observed. Thus, the new storm cells that will be observed will depend on the scanning action of the radar.

*The cost function $C(S, A, S) \to \mathcal{R}$* encodes the goals of the radar sensing network. $C$ is a function of the error between the true state and the observed state, whether all storms have been observed,

and a penalty term for not rescanning a storm within $T_r$ decision epochs. More precisely,

$$C \quad = \quad \sum_{i=1}^{N_p^o} \sum_{j=1}^{N_d} |d_{ij}^o - d_{ij}| + (N_p - N_p^o)P_m + \sum_{i=1}^{N_p} I(t_i)P_r \qquad (3)$$

where $N_p^o$ is the observed number of storms, $N_d$ is the number of attributes per storm, $d_{ij}^o$ is the observed value of attribute $j$ of storm $i$, $d_{ij}$ is the true value of attribute $j$ of storm $i$, $N_p$ is the true number of storms, $P_m$ is the penalty for missing a storm, $t_i$ is the number of decision epochs since storm $i$ was last scanned, $P_r$ is the penalty for not scanning a storm at least once within $T_r$ decision epochs, and $I(t_i)$ is an indicator function that equals 1 when $t_i \geq T_r$. The quality with which a storm is observed determines the difference between the observed and true values of its attributes.

We use linear Sarsa($\lambda$) [15] as the reinforcement learning algorithm to solve the MDP for the radar control problem. To obtain the basis functions, we use tile coding [13, 14]. Rather than defining tilings over the entire state space, we define a separate set of tilings for each of the state variables.

## 4 Evaluation

### 4.1 Simulation Environment

We consider radars with both 10 and 30km radii as in [5, 17]. Two overlapping radars are placed in a 90km $\times$ 60km rectangle, one at (30km, 30km) and one at (60km, 30km). A new storm cell can appear anywhere within the rectangle and a maximum number of cells can be present on any decision epoch. When the $(x, y)$ center of a storm cell is no longer within range of any radar, the cell is removed from the environment. Following [5], we use a 30-second decision epoch.

We derive the maximum storm cell radius from [11], which uses 2.83km as "the radius from the cell center within which the intensity is greater than $e^{-1}$ of the cell center intensity." We then permit a storm cell's radius to range from 1 to 4 km. To determine the range of storm cell velocities, we use 39 real storm cell tracks obtained from meteorologists. Each track is a series of $(latitude, longitude)$ coordinates. We first compute the differences in latitude and longitude, and in time, between successive pairs of points. We then fit the differences using Gaussian distributions. We obtain, in units of km/hour, that the latitude (or $x$) velocity has mean 9.1 km/hr and std. dev. of 35.6 km/hr and that the longitude (or $y$) velocity has mean 16.7 km/hr and std. dev. of 28.8 km/hr. To obtain a storm cell's $(x, y)$ velocity, we then sample the appropriate Gaussian distribution.

To simulate the environment transitions we use a stochastic model of rainfall in which storm cell arrivals are modeled using a spatio-temporal Poisson process, see [11, 1]. To determine the number of new storm cells to add during a decision epoch, we sample a Poisson random variable with rate $\lambda \eta \delta a \delta t$ with $\lambda = 0.075$ storm cells/$km^2$ and $\eta = 0.006$ storm cells/minute from [11]. From the radar setup we have $\delta a = 90 \cdot 60$ $km^2$, and from the 30-second decision epoch we have $\delta t = 0.5$ minutes. New storm cells are uniformly randomly distributed in the 90km $\times$ 60km region and we uniformly randomly choose new storm cell attributes from their range of values. This simulates the true state of the environment over time. The following simplified radar model determines how well the radars observe the true environmental state under a given set of radar configurations. If a storm cell $p$ is scanned using a set of radar configurations $S_r$, the location, velocity, and radius attributes are observed as a function of the $U_p(p, S_r)$ quality defined in Section 3.1. $U_p(p, S_r)$ returns a value $u$ between zero and one. Then the observed value of the attribute is the true value of the attribute plus some Gaussian noise distributed with mean zero and standard deviation $(1 - u)V^{max}/\rho$ where $V^{max}$ is the largest positive value the attribute can take and $\rho$ is a scaling term that will allow us to adjust the noise variability. Since $u$ depends on the decision epoch $t$, for the $k$-step look-ahead scan strategy we also use $\sigma_t = (1 - u_t)V^{max}/\rho$ to compute the measurement error covariance matrix, $R$, in our Kalman filter.

We parameterize the MDP cost function as follows. We assume that any unobserved storm cell has been observed with quality 0, hence $u = 0$. Summing over $(1 - u)V^{max}/\rho$ for all attributes with $\sigma = 0$ gives the value $P_m = 15.5667$, and thus a penalty of 15.5667 is received for each unobserved storm cell. If a storm cell is not seen within $T_r = 4$ decision epochs a penalty of $P_r = 200$ is given. Using the value 200 ensures that if a storm cell has not been rescanned within the appropriate amount of time, this part of the cost function will dominate.

We distinguish the true environmental state known only to the simulator from the observed environmental state used by the scan strategies for several reasons. Although radars provide measurements about meteorological phenomena, the true attributes of the phenomena are unknown. Poor overlap in a dual-Doppler area, scanning a subsector too quickly or slowly, or being unable to obtain a sufficient number of elevation scans will degrade the quality of the measurements. Consequently, models of previously existing phenomena may contain estimation errors such as incorrect velocity, propagating error into the future predicted locations of the phenomena. Additionally, when a radar scans a subsector, it obtains more accurate estimates of the phenomena in that subsector than if it had scanned a full $360°$, but less accurate estimates of the phenomena outside the subsector.

## 4.2 Results

In this section we present experimental results obtained using the simulation model of the previous section and the scan strategies described in Section 3. For the limited lookahead strategy we use $\beta = 0.5$, $\kappa_p = 0.25$, $\kappa_s = 0.25$, and $\phi = 0.75$. For Sarsa($\lambda$), we use a learning rate $\alpha = 0.0005$, exploration rate $\epsilon = 0.01$, discount factor $\gamma = 0.9$, and eligibility decay $\lambda = 0.3$. Additionally, we use a single tiling for each state variable. For the $(x, y)$ location and radius tilings, we use a granularity of 1.0; for the $(x, y)$ velocity, phenomenon confidence, and radar sector confidence tilings, we use a granularity of 0.1. When there are a maximum of four storms, we restrict Sarsa($\lambda$) to scanning only 180 or 360 degree sectors to reduce the time needed for convergence. Finally, all strategies are always compared over the same true environmental state.

Figure 2(a) shows an example convergence profile of Sarsa($\lambda$) when there are at most four storms in the environment. Figure 2(b) shows the average difference in scan quality between the learned Sarsa($\lambda$) strategy and sit-and-spin and 2-step strategies. When $1/\rho = 0.001$ (i.e., little measurement noise) Sarsa($\lambda$) has the same or higher relative quality than does sit-and-spin, but significantly lower relative quality (0.05 to 0.15) than does the 2-step. This in part reflects the difficulty of learning to perform as well as or better than Kalman filtering. Examining the learned strategy showed that when there was at most one storm with observation noise $1/\rho = 0.001$, Sarsa($\lambda$) learned to simply sit-and-spin, since sector scanning conferred little benefit. As the observation noise increases, the relative difference increases for sit-and-spin, and decreases for the 2-step. Figure 2(c) shows the average difference in cost between the learned Sarsa($\lambda$) scan strategy and the sit-and-spin and 2-step strategies for a 30 km radar radius. Sarsa($\lambda$) has the lowest average cost.

Looking at the Sarsa($\lambda$) inter-scan times, Figure 2 (d) shows that, as a consequence of the penalty for not scanning a storm within $T_r = 4$ time-steps, while Sarsa($\lambda$) may rescan fewer storm cells within 1, 2, or 3 decision epochs than do the other scan strategies, it scans almost all storm cells within 4 epochs. Note that for the sit-and-spin CDF, $P[X \leq 1]$ is not 1; due to noise, for example, the measured location of a storm cell may be (expected) outside any radar footprint and consequently the storm cell will not be observed. Thus the 2-step has more inter-scan times greater than $T_r = 4$ than does Sarsa($\lambda$). Together with Figure 2(b) and (c), this implies that there is a trade-off between inter-scan time and scan quality. We hypothesize that this trade-off occurs because increasing the size of the scan sectors ensures that inter-scan time is minimized, but decreases the scan quality.

Other results (not shown, see [7]) examine the average difference in quality between the 1-step and 2-step strategies for 10 km and 30 km radar radii. With a 10 km radius, the 1-step quality is essentially the same as the 2-step quality. We hypothesize that this is a consequence of the maximum storm cell radius, 4 km, relative to the 10 km radar radius. With a 30 km radius and at most eight storm cells, the 2-step quality is about 0.005 better than the 1-step and about 0.07 better than sit-and-spin (recall that quality is a value between 0 and 1). Now recall that Figure 2(b) shows that with a 30 km radius and at most four storm cells, the 2-step quality is as much as 0.12 than sit-and-spin. This indicates that there may be some maximum number of storms above which it is best to sit-and-spin.

Overall, depending on the environment in which the radars are deployed, there are decreasing marginal returns for considering more than 1 or 2 future expected states. Instead, the primary value of reinforcement learning for the radar control problem is balancing multiple conflicting goals, i.e., maximizing scan quality while minimizing inter-scan time. Implementing the learned reinforcement learning scan strategy in a real meteorological radar network requires addressing the differences between the offline environment in which the learned strategy is trained, and the online environment in which the strategy is deployed. Given the slow convergence time for Sarsa($\lambda$) (on the order of

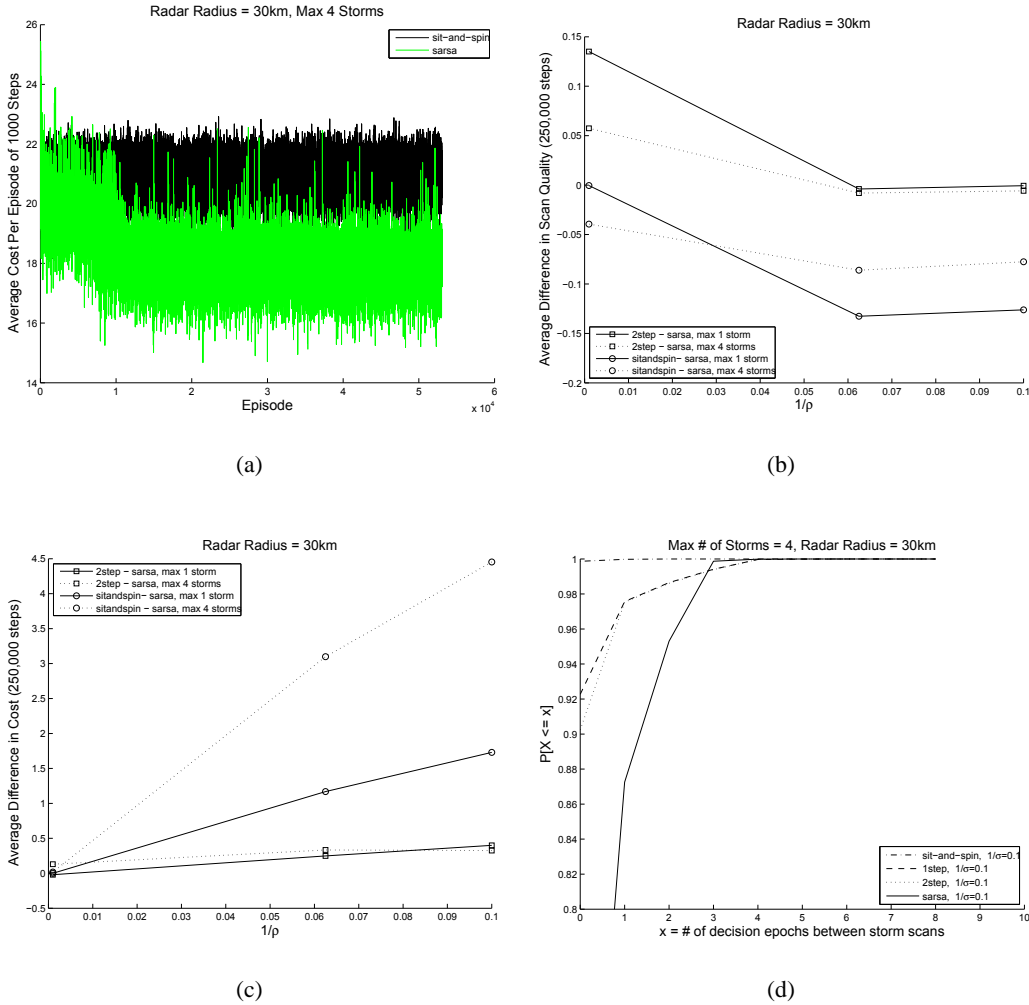

Figure 2: Comparing the scan strategies based on quality, cost, and inter-scan time. Recall that $\rho$ is a scaling term used to determine measurement noise, see Section 4.1.

days), training solely online is likely infeasible, although the time complexity could be mitigated by using hierarchical reinforcement learning methods and semi-Markov decision process. Some online training could be achieved by treating $360°$ scans as the true environment state. Then when unknown states are entered, learning could be performed, alternating between $360°$ scans to gauge the true state of the environment and exploratory scans by the reinforcement learning algorithm.

## 5 Related Work

Other reinforcement learning applications in large state spaces include robot soccer [12] and helicopter control [10]. With respect to radar control, [4] examines the problem of using agile radars on airplanes to detect and track ground targets. They show that lookahead scan strategies for radar tracking of a ground target outperform myopic strategies. In comparison, we consider the problem of tracking meteorological phenomena using ground radars. [4] uses an information theoretic measure to define the reward metric and proposes both an approximate solution to solving the MDP Bellman equations as well as a Q-learning reinforcement learning-based solution. [16] examines where to target radar beams and which waveform to use for electronically steered phased array radars. They maintain a set of error covariance matrices and dynamical models for existing targets, as well as

track existence probability density functions to model the probability that targets appear. They then choose the scan mode for each target that has both the longest revisit time for scanning a target and error covariance below a threshold. They do this for control 1-step and 2-steps ahead and show that considering the environment two decision epochs ahead outperforms a 1-step look-ahead for tracking of multiple targets.

# 6    Conclusions and Future Work

In this work we compared the performance of myopic and lookahead scan strategies in the context of the meteorological radar control problem. We showed that the main benefits of using a lookahead strategy are when there are multiple meteorological phenomena in the environment, and when the maximum radius of any phenomenon is sufficiently smaller than the radius of the radars. We also showed that there is a trade-off between the average quality with which a phenomenon is scanned and the number of decision epochs before which a phenomenon is rescanned. Overall, considering only scan quality, a simple lookahead strategy is sufficient. To additionally consider inter-scan time (or optimize over multiple metrics of interest), a reinforcement learning strategy is useful. For future work, rather than identifying a policy that chooses the best action to execute in a state for a single decision epoch, it may be useful to consider actions that cover multiple epochs, as in semi-Markov decision processes or to use controllers from robotics [3]. We would also like to incorporate more radar and meteorological information into the transition, measurement, and cost functions.

### Acknowledgments

The authors thank Don Towsley for his input. This work was supported in part by the National Science Foundation under the Engineering Research Centers Program, award number EEC-0313747. Any opinions, findings and conclusions or recommendations expressed in this material are those of the author(s) and do not necessarily reflect those of the National Science Foundation.

## References

[1] D. Cox and V. Isham. A simple spatial-temporal model of rainfall. *Proceedings of the Royal Society of London. Series A, Mathematical and Physical Sciences*, 415:1849:317–328, 1988.

[2] B. Donovan and D. J. McLaughlin. Improved radar sensitivity through limited sector scanning: The DCAS approach. In *Proceedings of AMS Radar Meteorology*, 2005.

[3] M. Huber and R. Grupen. A feedback control structure for on-line learning tasks. *Robotics and Autonomous Systems*, 22(3-4):303–315, 1997.

[4] C. Kreucher and A. O. H. III. Non-myopic approaches to scheduling agile sensors for multistage detection, tracking and identification. In *Proceedings of ICASSP*, pages 885–888, 2005.

[5] J. Kurose, E. Lyons, D. McLaughlin, D. Pepyne, B. Phillips, D. Westbrook, and M. Zink. An end-user-responsive sensor network architecture for hazardous weather detection, prediction and response. *AINTEC*, 2006.

[6] C. Kwok and D. Fox. Reinforcement learning for sensing strategies. In *IROS*, 2004.

[7] V. Manfredi and J. Kurose. Comparison of myopic and lookahead scan strategies for meteorological radars. Technical Report U of Massachusetts Amherst, 2006-62, 2006.

[8] V. Manfredi, S. Mahadevan, and J. Kurose. Switching kalman filters for prediction and tracking in an adaptive meteorological sensing network. In *IEEE SECON*, 2005.

[9] K. Murphy. A survey of POMDP solution techniques. Technical Report U.C. Berkeley, 2000.

[10] A. Ng, A. Coates, M. Diel, V. Ganapathi, J. Schulte, B. Tse, E. Berger, and E. Liang. Inverted autonomous helicopter flight via reinforcement learning. In *International Symposium on Experimental Robotics*, 2004.

[11] I. Rodrigues-Iturbe and P. Eagleson. Mathematical models of rainstorm events in space and time. *Water Resources Research*, 23:1:181–190, 1987.

[12] P. Stone, R. Sutton, and G. Kuhlmann. Reinforcement learning for robocup-soccer keepaway. *Adaptive Behavior*, 3, 2005.

[13] R. Sutton. Tile coding software. *http://rlai.cs.ualberta.ca/RLAI/RLtoolkit/tiles.html*.

[14] R. Sutton. Generalization in reinforcement learning: Successful examples using sparse coarse coding. In *NIPS*, 1996.

[15] R. Sutton and A. Barto. *Reinforcement Learning: An Introduction*. MIT Press, Cambridge, Massachusetts, 1998.

[16] S. Suvorova, D. Musicki, B. Moran, S. Howard, and B. L. Scala. Multi step ahead beam and waveform scheduling for tracking of manoeuvering targets in clutter. In *Proceedings of ICASSP*, 2005.

[17] J. M. Trabal, B. C. Donovan, M. Vega, V. Marrero, D. J. McLaughlin, and J. G. Colom. Puerto Rico student test bed applications and system requirements document development. In *Proceedings of the 9th International Conference on Engineering Education*, 2006.

